# NEURAL ARCHITECTURE

Valentino Braitenberg
Max Planck Institute
Federal Republic of Germany

While we are waiting for the ultimate biophysics of cell membranes and synapses to be completed, we may speculate on the shapes of neurons and on the patterns of their connections. Much of this will be significant whatever the outcome of future physiology.

Take as an example the isotropy, anisotropy and periodicity of different kinds of neural networks. The very existence of these different types in different parts of the brain (or in different brains) defeats explanation in terms of embryology; the mechanisms of development are able to make one kind of network or another. The reasons for the difference must be in the functions they perform. The tasks which they solve in one case apparently refer to some space which is intrinsically isotropic, in another to a situation in which different coordinates mean different things. In the periodic case, the tasks obviously refer to some kind of modules and to their relations.

The examples I have in mind are first the cerebral cortex, quite isotropic in the plane of the cortex, second the cerebellar cortex with very different sets of fibers at right angles to each other (one excitatory, as we know today, and the other inhibitory) and third some of the nerve nets behind the eye of the fly.

Besides general patterns of symmetry, some simple statements of a statistical nature can be read off the histological picture. If a neuron is a device picking up excitation (and/or inhibition) on its ten to ten thousand afferent synapses and producing excitation (or inhibition) on ten to ten thousand synapses on other neurons, the density, geometrical distribution and reciprocal overlap of the clouds of afferent and of efferent synapses of individual neurons provide unquestionable constraints to neural computation. In the simple terms of the histological practitioner, this translates into the description of the shapes of dendritic and axonal trees, into counts of neurons and synapses, differential counts of synapses of the excitatory and inhibitory kind and measurements of the axonal and dendritic lengths.